# Margin Analysis of the LVQ Algorithm

**Koby Crammer**
*kobics@cs.huji.ac.il*

**Ran Gilad-Bachrach**
*ranb@cs.huji.ac.il*

**Amir Navot**
*anavot@cs.huji.ac.il*

**Naftali Tishby**
*tishby@cs.huji.ac.il*
School of Computer Science and Engineering and
Interdisciplinary Center for Neural Computation
The Hebrew University, Jerusalem, Israel

## Abstract

Prototypes based algorithms are commonly used to reduce the computational complexity of Nearest-Neighbour (NN) classifiers. In this paper we discuss theoretical and algorithmical aspects of such algorithms. On the theory side, we present margin based generalization bounds that suggest that these kinds of classifiers can be more accurate then the 1-NN rule. Furthermore, we derived a training algorithm that selects a good set of prototypes using large margin principles. We also show that the 20 years old Learning Vector Quantization (LVQ) algorithm emerges naturally from our framework.

## 1   Introduction

Though fifty years have passed since the introduction of One Nearest Neighbour (*1-NN*) [1] it is still a popular algorithm. 1-NN is a simple and intuitive algorithm but at the same time achieves state of the art results [2]. However in large, high dimensional data set it often become infeasible. One approach to face this computational problem is to approximate the nearest neighbour [3] using various techniques. Alternative approach is to choose a small data-set (aka prototypes) which represents the original training sample, and apply the nearest neighbour rule only with respect to this small data-set. This solution maintains the "spirit" of the original algorithm, while making it feasible. Moreover, it might improve the accuracy by reducing noise over-fitting.

In this setting, the goal of the learning stage is to choose wisely the prototypes, i.e., in a way that will yield good generalization [1]. In this paper we use the *Maximal Margin* principle [4, 5] for this purpose. The training data is used to measure the margin of each proposed positioning of the prototypes. We combine these measurements to calculate a risk for each prototype set and select the prototypes that minimize the risk.

Roughly speaking, margins measure the level of confidence a classifiers has with respect to its decisions. This tool has become a primary method in machine learning during the last decade. Two of the most powerful algorithms in the field, *Support Vector Machines*

(SVM) [4] and *AdaBoost* [5] are motivated and analyzed by margins. Since the introduction of these algorithms dozens of papers were published on different aspect of margins in supervised learning [6, 7, 8].

*Learning Vector Quantization* (LVQ) [9] is a well-known algorithm that deals with the same problem of selecting prototypes. LVQ iterates over the training data and updates the prototypes position. Although it is known for more then 20 years and in spite of its popularity, no adequate generalization bounds and theory were suggested for this algorithm. In this paper we show that algorithms derived from the maximal margin principle contains LVQ as a special case. We use this result to present generalization bounds and insights for the LVQ algorithm.

Buckingham and Geva [10] were the first to explore the relations between maximal margin principle and LVQ. They presented a variant named LMVQ and analyzed it. As in most of the literature about LVQ they look at the algorithm as trying to estimate a density function (or a function of the density) at each point. After estimating the density the Bayesian decision rule is used. We take a different point of view on the problem and look at the geometry of the decision boundary induced by the decision rule. Note that in order to generate a good classification rule the only significant factor is where the decision boundary lies (It is a well known fact that classification is easier then density estimation [11]).

**Summary of the Results**   In section 2 we present the model and outline the LVQ family of algorithms. A discussion and definition of margin is provided in section 3. The two fundamental results are a bound on the generalization error and a theoretical reasoning for the LVQ family of algorithms. In section 4 we present a bound on the gap between the empirical and the generalization accuracy. This provides a guaranty on the performance over unseen instances based on the empirical evidence. Although LVQ was designed as an approximation to nearest neighbour the theorem suggests that the former is more accurate in many cases. Indeed a simple experiment shows this prediction to be true. In section 5 we show how LVQ family of algorithms emerges from the generalization bound. These algorithms minimize the bound using gradient descent. The different variants correspond to different tradeoff between opposing quantities. In practice the tradeoff is controlled by *l*oss functions.

## 2   Problem Setting and the LVQ algorithm

The framework we are interested in is supervised learning for classification problems. In this framework the task is to find a map from $\mathbb{R}^n$ into a finite set of labels $\mathcal{Y}$. We focus on classification functions of the following form: the classifiers are parameterized by a set of points $\mu_1, \ldots, \mu_k \in \mathbb{R}^n$ which we refer to as *prototypes*. Each prototype is associated with a label $y \in \mathcal{Y}$. Given a new instance $x \in \mathbb{R}^n$ we predict that it has the same label as the closest prototype, similar to the 1-nearest-neighbour rule (1-NN). We denote the label predicted using a set of prototypes $\{\mu_j\}_{j=1}^k$ by $\mu(x)$. The goal of the learning process in this model is to find a set of prototypes which will predict accurately the labels of unseen instances.

The Learning Vector Quantization (LVQ) family of algorithms works in this model. The algorithm gets as an input a labelled sample $S = \{(x_l, y_l)\}_{l=1}^m$, where $x_l \in \mathbb{R}^n$ and $y_l \in \mathcal{Y}$ and uses it to find a good set of prototypes. All the variants of LVQ share the following common scheme. The algorithm maintains a set of prototypes each is assigned with a predefined label, which is kept constant during the learning process. It cycles through the training data $S$ and on each iteration modifies the set of prototypes in accordance to one instance $(x_t, y_t)$. If the prototype $\mu_j$ has the same label as $y_t$ it is attracted to $x_t$ but if the label of $\mu_j$ is different it is repelled from it. Hence LVQ updates the closest prototypes to

$x_t$ according to the rule:

$$\mu_j \leftarrow \mu_j \pm \alpha_t(x_t - \mu_j) \,, \tag{1}$$

where the sign is positive if the label of $x_t$ and $\mu_j$ agree, and negative otherwise. The parameter $\alpha_t$ is updated using a predefined scheme and controls the rate of convergence of the algorithm. The variants of LVQ differ in which prototypes they choose to update in each iteration and in the specific scheme used to modify $\alpha_t$.

For instance, LVQ1 and OLVQ1 updates only the closest prototype to $x_t$ in each iteration. Another example is the LVQ2.1 which modifies the two closest prototypes $\mu_i$ and $\mu_j$ to $x_t$. It uses the same update rule (1) but apply it only if the following two conditions hold :

1. Exactly one of the prototypes has the same label as $x_t$, i.e. $y_t$.
2. The ratios of their distances from $x_t$ falls in a window: $1/s \leq \|x_t - \mu_i\| / \|x_t - \mu_j\| \leq s$, where $s$ is the window size.

More variants of LVQ can be found in [9].

## 3  Margins

Margin plays an important role in current research of machine learning. It measures the confidence of a classifier with respect to its predictions. One approach is to define margin as the distance between an instance and the decision boundary induced by the classification rule as illustrated in figure 1(a). Support Vector Machines [4] are based on this definition of margin, which we refer to as *Sample-Margin*. However, an alternative definition, *Hypothesis Margin*, exists. In this definition the margin is the distance that the classifier can travel without changing the way it labels any of the sample points. Note that this definition requires a distance measure between classifiers. This type of margin is used in AdaBoost [5] and is illustrated in figure 1(b).

It is possible to apply these two types of margin in the context of LVQ. Recall that in our model a classifier is defined by a set of labeled prototypes. Such a classifier generates a decision boundary by Voronoi tessellation. Although using sample margin is more natural as a first choice, it turns out that this type of margin is both hard to compute and numerically unstable in our context, since small relocations of the prototypes might lead to a dramatic change in the sample margin. Hence we focus on the hypothesis margin and thus have to define a distance measure between two classifiers. We choose to define it as the maximal distance between prototypes pairs as illustrated in figure 2. Formally, let $\mu = \{\mu_j\}_{j=1}^k$ and $\hat{\mu} = \{\hat{\mu}_j\}_{j=1}^k$ define two classifiers, then

$$\rho(\mu, \hat{\mu}) = \max_{i=1}^{k} \|\mu_i - \hat{\mu}_i\|_2 \ .$$

Note that this definition is not invariant to permutations of the prototypes but it upper bounds the invariant definition. Furthermore, the induced margin is easy to compute (lemma 1) and lower bounds the sample-margin (lemma 2).

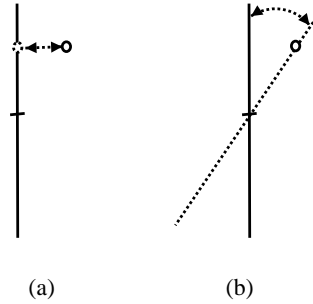

(a)         (b)

Figure 1: *Sample Margin* (figure 1(a)) measures how much can an **instance** travel before it hits the decision boundary. On the other hand *Hypothesis Margin* (figure 1(b)) measures how much can the **hypothesis** travel before it hits an instance.

**Lemma 1** *Let $\mu = \{\mu_j\}_{j=1}^k$ be a set of prototypes and $x$ a sample point. Then the hypothesis margin of $\mu$ with respect to $x$ is $\theta = \frac{1}{2}(\|\mu_j - x\| - \|\mu_i - x\|)$ where $\mu_i$ ($\mu_j$) is the closest prototype to $x$ with the same (alternative) label.*

**Lemma 2** *Let $S = \{x_l\}_{l=1}^m$ be a sample and $\mu = (\mu_1, \ldots, \mu_k)$ be a set of prototypes.*

$$\text{sample-margin}_S(\mu) \geq \text{hypothesis-margin}_S(\mu)$$

Lemma 2 shows that if we find a set of prototypes with large hypothesis margin then it has large sample margin as well.

## 4   Margin Based Generalization Bound

In this section we present a bound on the generalization error of LVQ type of classifiers.

When a classifier is applied to a training data it is natural to use the training error as a prediction to the generalization error (the probability of misclassification of an unseen instance). In prototype based hypothesis the classifier assigns a confidence level, i.e. margin, to its predictions. Taking into account the margin by counting instances with small margin as mistakes gives a better prediction and provide a bound on the generalization error. This bound is given in terms of the number of prototypes, the sample size, the margin and the margin based empirical error. The following theorem states this result formally.

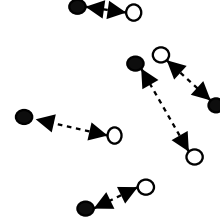

Figure 2: The distance measure on the LVQ hypothesis class. The distance between the white and black prototypes set is the maximal distance between prototypes pairs.

**Theorem 1** *In the following setting:*

- *Let $S = \{x_i, y_i\}_{i=1}^m \in \{\mathbb{R}^n \times \mathcal{Y}\}^m$ be a training sample drawn by some underlying distribution $\mathcal{D}$.*
- *Assume that $\forall i \quad \|x_i\| \leq R$.*
- *Let $\mu$ be a set of prototypes with $k$ prototypes from each class.*
- *Let $0 < \theta < 1/2$.*
- *Let $\alpha_S^\theta(\mu) = \frac{1}{m} \left| \{i : margin_\mu(x_i) < \theta\} \right|$.*
- *Let $e_\mathcal{D}(\mu)$ be the generalization error: $e_\mathcal{D}(\mu) = Pr_{(x,y)\sim\mathcal{D}} [\mu(x) \neq y]$.*
- *Let $\delta > 0$.*

*Then with probability $1 - \delta$ over the choices of the training data:*

$$\forall \mu \qquad e_\mathcal{D} \leq \alpha_S^\theta(\mu) + \sqrt{\frac{8}{m}\left(d\log^2\frac{32m}{\theta^2} + \log\frac{4}{\delta}\right)} \qquad (2)$$

*where $d$ is the VC dimension:*

$$d = \min\left(n+1, \frac{64R^2}{\theta^2}\right) 2k^{|\mathcal{Y}|}\log ek^2 \qquad (3)$$

This theorem leads to a few observations. First, note that the bound is dimension free, in the sense that the generalization error is bounded independently of the input dimension ($n$) much like in SVM. Hence it makes sense to apply these algorithms with kernels.

Second, note that the VC dimension grows as the number of prototypes grows (3). This suggest that using too many prototypes might result in poor performance, therefore there

is a non trivial optimal number of prototypes. One should not be surprised by this result as it is a realization of the *Structural Risk Minimization* (SRM) [4] principle. Indeed a simple experiment supports this prediction. Hence not only that prototype based methods are faster than *Nearest Neighbour*, they are more accurate as well. Due to space limitations proofs are provided in the full version of this paper only.

## 5   Maximizing Hypothesis Margin Through Loss Function

Once margin is properly defined it is natural to ask for algorithm that maximizes it. We will show that this is exactly what LVQ does. Before going any further we have to understand why maximizing the margin is a good idea.

In theorem 1 we saw that the generalization error can be bounded by a function of the margin $\theta$ and the empirical $\theta$-error ($\alpha$). Therefore it is natural to seek prototypes that obtain small $\theta$-error for a large $\theta$. We are faced with two contradicting goals: small $\theta$-error verses large $\theta$. A natural way to solve this problem is through the use of loss function.

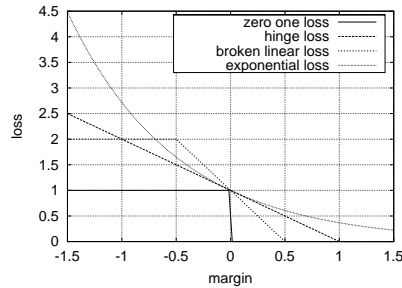

Loss function are a common technique in machine learning for finding the right balance between opposed quantities [12]. The idea is to associate a margin based loss (a "cost") for each hypothesis with respect to a sample. More formally, let $L$ be a function such that:

1.   For every $\theta$:        $L(\theta) \geq 0$.
2.   For every $\theta < 0$:    $L(\theta) \geq 1$.

Figure 3: Different loss functions. SVM, LVQ1 and OLVQ1 use the "hinge" loss: $(1 - \theta)_+$. LVQ2.1 uses the broken linear: $\min(2, (1 - 2\theta)_+)$. AdaBoost use the exponential loss ($e^{-\theta}$).

We use $L$ to compute the loss of an hypothesis with respect to one instance. When a training set is available we sum the loss over the instances: $\mathcal{L}(\mu) = \sum_l L(\theta_l)$, where $\theta_l$ is the margin of the $l$'th instance in the training data. The two axioms of loss functions guarantee that $\mathcal{L}(\mu)$ bounds the empirical error. It is common to add more restrictions on the loss function, such as requiring that $L$ is a non-increasing function. However, the only assumption we make here is that the loss function $L$ is differentiable.

Different algorithms use different loss functions [12]. AdaBoost uses the exponential loss function $L(\theta) = e^{-\beta\theta}$ while SVM uses the "hinge" loss $L(\theta) = (1 - \beta\theta)_+$, where $\beta > 0$ is a scaling factor. See figure 3 for a demonstration of these loss functions.

Once a loss function is chosen, the goal of the learning algorithm is finding an hypothesis that minimizes it. Gradient descent is a natural simple choice for the task. Recall that in our case $\theta_l = (\|x_l - \mu_i\| - \|x_l - \mu_j\|)/2$ where $\mu_j$ and $\mu_i$ are the closest prototypes to $x_l$ with the correct and incorrect labels respectively. Hence we have that[2]

$$\frac{d\theta_l}{d\mu_r} = S_l(r)\frac{x_l - \mu_r}{\|x_l - \mu_r\|}$$

where $S_l(r)$ is a sign function such that

$$S_l(r) = \begin{cases} 1 & \text{if } \mu_r \text{ is the closest prototype with correct label.} \\ -1 & \text{if } \mu_r \text{ is the closest prototype with incorrect label.} \\ 0 & \text{otherwise.} \end{cases}$$

**Algorithm 1** Online Loss Minimization.
Recall that $L$ is a loss function, and $\gamma_t$ varies to zero as the algorithm proceeds.
---

1. Choose an initial positions for the prototypes $\{\mu_j\}_{j=1}^k$.

2. For $t = 1 : T($ or $\infty)$

    (a) Receive a labelled instance $x_t, y_t$

    (b) Compute the closest correct and incorrect prototypes to $x_t$: $\mu_j, \mu_i$, and the margin of $x_t$, i.e. $\theta_t = 1/2(\|x_t - \mu_i\| - \|x_t - \mu_j\|)$

    (c) Apply the update rule for $r = i, j$:

    $$\mu_r \leftarrow \mu_r + \gamma_t \frac{dL(\theta_t)}{d\theta} S_l(r) \frac{x_t - \mu_r}{\|x_t - \mu_r\|}$$

---

Taking the derivative of $\mathcal{L}$ with respect to $\mu_r$ using the chain rule we obtain

$$\frac{d\mathcal{L}}{d\mu_r} = \sum_l \frac{dL(\theta_l)}{d\theta_l} S_l(r) \frac{x_l - \mu_r}{\|x_l - \mu_r\|} \tag{4}$$

By comparing the derivative to zero, we get that the optimal solution is achieved when $\mu_r = \sum_l w_l^r x_l$ where $\alpha_l^r = \frac{dL(\theta_l)}{d\theta_l} \frac{S_l(r)}{\|x_l - \mu_r\|}$ and $w_l^r = \frac{\alpha_l^r}{\sum_l \alpha_l^r}$. This leads to two conclusions. First, the optimal solution is in the span of the training instances. Furthermore, from its definition it is clear that $w_l^r \neq 0$ only for the closest prototypes to $x_l$. In other words, $w_l^r \neq 0$ if and only if $\mu_r$ is either the closest prototype to $x_l$ which have the same label as $x_l$, or the closest prototype to $x_l$ with alternative label. Therefore the notion of support vectors [4] applies here as well.

## 5.1 Minimizing The Loss

Using (4) we can find a local minima of the loss function by a gradient descent algorithm. The iteration in time $t$ computes:

$$\mu_r(t+1) \leftarrow \mu_r(t) + \gamma_t \sum_l \frac{dL(\theta_l)}{d\theta} S_l(r) \frac{x_l - \mu_r(t)}{\|x_l - \mu_r(t)\|}$$

where $\gamma_t$ approaches zero as $t$ increases. This computation can be done iteratively where in each step we update $\mu_r$ only with respect to one sample point $x_l$. This leads to the following basic update step

$$\mu_r \leftarrow \mu_r + \gamma_t \frac{dL(\theta_l)}{d\theta} S_l(r) \frac{x_l - \mu_r}{\|x_l - \mu_r\|}$$

Note that $S_l(r)$ differs from zero only for the closest correct and incorrect prototypes to $x_l$, therefore a simple online algorithm is obtained and presented as algorithm 1.

## 5.2 LVQ1 and OLVQ1

The online loss minimization (algorithm 1) is a general algorithm applicable with different choices of loss functions. We will now apply it with a couple of loss functions and see how LVQ emerges. First let us consider the "hinge" loss function. Recall that the hinge loss is defined to be $L(\theta) = (1 - \beta\theta)_+$. The derivative[3] of this loss function is

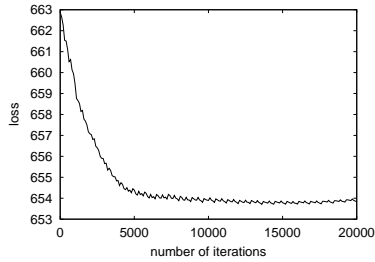

$$\frac{dL(\theta)}{d\theta} = \begin{cases} 0 & \text{if } \theta > 1/\beta \\ -\beta & \text{otherwise} \end{cases}$$

If $\beta$ is chosen to be large enough, the update rule in the online loss minimization is

$$\mu_r = \mu_r \pm \gamma_t \beta \frac{x_t - \mu_r}{\|x_t - \mu_r\|}$$

Figure 4: The "hinge" loss function ($\sum (1 - \theta_l)_+$) vs. number of iterations of OLVQ1. One can clearly see that it decreases.

This is the same update rule as in LVQ1 and OLVQ1 algorithm [9] beside the extra factor of $\frac{\beta}{\|x_t - \mu_r\|}$. However, this is a minor difference since $\beta / \|x_t - \mu_r\|$ is just a normalizing factor. A demonstration of the affect of OLVQ1 on the "hinge" loss function is provided in figure 4. We applied the algorithm to a simple toy problem consisting of three classes and a training set of 800 points. We allowed the algorithm 10 prototypes. As expected the loss decreases as the algorithm proceeds. For this purpose we used the lvq_pak package [13].

### 5.3   LVQ2.1

The idea behind the definition of margin, and especially hypothesis margin was that a minor change in the hypothesis can not change the way it labels an instance which had a large margin. Hence when making small updates (i.e. small $\gamma_t$) one should focus only on the instances which have margins close to zero. The same idea appeared also in Freund's boost by majority algorithm [14].

Kohonen adapted this idea to his LVQ2.1 algorithm [9]. The major difference between LVQ1 and LVQ2.1 algorithm is that LVQ2.1 updates $\mu_r$ only if the margin of $x_t$ falls inside a certain window. The suitable loss function for LVQ2.1 is the broken linear loss function (see figure 3). The broken linear loss is defined to be $L(\theta) = \min(2, (1 - \beta\theta)_+)$. Note that for $|\theta| > 1/\beta$ the loss is constant (i.e. the derivative is zero), this causes the learning algorithm to overlook instances with too high or too low margin. There exist several differences between LVQ2.1 and the online loss minimization presented here, however these differences are minor.

## 6   Conclusions and Further Research

In this paper we used the maximal margin principle together with loss functions to derive algorithms for prototype positioning. We saw that LVQ can be considered as a special case of this general algorithm. We also provide generalization bounds for any prototype based classifier.

This formulation allows derivation of new algorithms in several different ways. The first is to use other loss functions such as the exponential loss. A second way is to use other classification rule, such as $k$-NN or parzan window. The proper way to adapt the algorithm to the chosen rule is to define the margin accordingly, and modify the minimization process in the training stage. We have constructed some basic experiments using the $k$-NN rule. The performance of the modified classifier did not exceed those of the 1-NN rule. We suggest the following explanation of these results. Usually the $k$-NN rule perform better than the 1-NN rule as it filters noise better, and in our setting the noise filtering is already achieved by using a small number of prototype.

Another extension to use a different distance measure instead of the $l_2$ norm. This may result in more complicated formula of the derivative of the loss function, but may improve the results significantly in some cases. One specific interesting distance measure is the Tangent Distance [2].

We also presented a generalization guarantee for prototype based classifier that is based on the margin training error. The bound is dimension free and thus a kernel version of the algorithm may yield a good performance. This modification is straightforward, as the algorithm can be expressed as function of inner-products only. We performed preliminary experiments with a kernelized version of the algorithm. It seems that it improves the accuracy when it is used with a small number of prototypes. However, allowing more prototypes to the standard version achieves the same improvement.

A possible explanation of this phenomenon is the following. Recall that a classifier is parametrised by a set of labelled prototypes that define a Voronoi tessellation. The decision boundary of such a classifier is built of some of the lines of the Voronoi tessellation. In the standard version these lines are straight lines. In the kernel version these lines are smooth non-linear curves. As the number of prototypes grows, the decision boundary consists of more, and shorter lines. Now, if we remember the fact that any smooth curve can be approximated by a broken linear line, we come to the conclusion that any classifier that can be generated by the kernel version, can be approximated by one that is generated by the standard version, when is applied with more prototypes.

**Acknowledgement**    We thank Yoram Singer and Gal Chechik for their helpful remarks.

## Footnotes

[1]Good generalization means that the probability of misclassifying a new example is small.

[2]Note that if $x_l = \mu_j$ the derivative is not defined. This extreme case does not affect our conclusions, hence or the sake of clarity we avoid the treatment of such extreme cases in this paper.

[3]The "hinge" loss has no derivative at the point $\theta = 1/\beta$. Again as in other cases in this paper, this fact is neglected.

# References

[1] E. Fix and j. Hodges. Discriminatory analysis. nonparametric discrimination: Consistency properties. Technical Report 4, USAF school of Aviation Medicine, 1951.

[2] P. Y. Simard, Y. A. Le Cun, and J. Denker. Efficient pattern recognition using a new transformation distance. In *Advances in Neural Information Processing Systems*, volume 5, pages 50–58. 1993.

[3] P. Indyk and R. Motwani. Approximate nearest neighbors: towards removing the curse of dimensionality. In *Proceedings of the 30th ACM Symposium on the Theory of Computing*, pages 604–613, 1998.

[4] V. Vapnik. *The Nature Of Statistical Learning Theory*. Springer-Verlag, 1995.

[5] Y. Freund and R. E. Schapire. A decision-theoretic generalization of on-line learning and an application to boosting. *Journal of Computer and System Sciences*, 55(1):119–139, 1997.

[6] R. E. Schapire, Y. Freund, P. Bartlett, and W. S. Lee. Boosting the margin : A new explanation for the effectiveness of voting methods. *Annals of Statistics*, 1998.

[7] Llew Mason, P. Bartlett, and J. Baxter. Direct optimization of margins improves generalization in combined classifier. *Advances in Neural Information Processing Systems*, 11:288–294, 1999.

[8] C. Campbell, N. Cristianini, and A. Smola. Query learning with large margin classifiers. In *International Conference on Machine Learning*, 2000.

[9] T. Kohonen. *Self-Organizing Maps*. Springer-Verlag, 1995.

[10] L. Buckingham and S. Geva. Lvq is a maximum margin algorithm. In *Pacific Knowledge Acquisition Workshop PKAW'2000*, 2000.

[11] L. Devroye, L. Gyorfi, and G. Lugosi. *A Probabilistic Theory of Pattern Recognition*. Springer, New York, 1996.

[12] Y. Singer and D. D. Lewis. Machine learning for information retrieval: Advanced techniques. presented at ACM SIGIR 2000, 2000.

[13] T. Kohonen, J. Hynninen, J. Kangas, and K. Laaksonen, J. Torkkola. Lvq_pak, the learning vector quantization program package. http://www.cis.hut.fi /research/lvq_pak, 1995.

[14] Y. Freund. Boosting a weak learning algorithm by majority. *Information and Computation*, 121(2):256–285, 1995.
